# The Cocktail Party Problem: Speech/Data Signal Separation Comparison between Backpropagation and SONN

John Kassebaum
jak@ec.ecn.purdue.edu

Manoel Fernando Tenorio
tenorio@ee.ecn.purdue.edu

Christoph Schaefers

Parallel Distributed Structures Laboratory
School of Electrical Engineering
Purdue University
W. Lafayette, IN. 47907

## ABSTRACT

This work introduces a new method called Self Organizing Neural Network (SONN) algorithm and compares its performance with Back Propagation in a signal separation application. The problem is to separate two signals; a modem data signal and a male speech signal, added and transmitted through a 4 khz channel. The signals are sampled at 8 khz, and using supervised learning, an attempt is made to reconstruct them. The SONN is an algorithm that constructs its own network topology during training, which is shown to be much smaller than the BP network, faster to trained, and free from the trial-and-error network design that characterize BP.

## 1. INTRODUCTION

The research in Neural Networks has witnessed major changes in algorithm design focus, motivated by the limitations perceived in the algorithms available at the time. With the extensive work performed in that last few years using multilayered networks, it was soon discovered that these networks present limitations in tasks

that: (a) are difficult to determine problem complexity a priori, and thus design network of the correct size, (b) training not only takes prohibitively long times, but requires a large number of samples as well as fine parameter adjustment, without guarantee of convergence, (c) such networks do not handle the system identification task efficiently for systems whose time varying structure changes radically, and, (d) the trained network is little more than a black box of weights and connections, revealing little about the problem structure; being hard to find the justification for the algorithm weight choice, or an explanation for the output decisions based on an input vector. We believe that this need is sparking the emergence of a third generation of algorithms to address such questions.

## 2. THE SELF ORGANIZING NEURAL NETWORK ALGORITHM

### 2.1 SELF ORGANIZING NETWORK FAMILY

A family of Self Organizing Structure (SOS) Algorithms can be readily designed with our present knowledge, and can be used as a tool to research the motivating questions. Each individual algorithm in this family might have different characteristics, which are summarized in the following list:

- A search strategy for the structure of the final model

- A rule of connectivity

- A performance criteria

- A transfer function set with appropriate training rule

As we will show here, by varying each one of these components, a different behavior of the algorithm can be imposed.

Self organizing structure algorithms are not new. These algorithms have been present in the statistical literature since the mid 70's in a very different context. As far as we know, the first one to propose such an algorithm was Ivahnenko [1971] which was followed by a host of variations on that original proposal [Duffy&Franklin, 1975; Ikeda, et al., 1976; Tomura&Kondo, 1980; Farlow,1989]. Ivahnenko's subfamily of algorithms (GMDH - Group Method of Data Handling) can be characterized in our classification by the same four-tuple criterion: (1) gradient descent local search, (2) creation of regular feedforward layers with elements pairwisely connected, (3) least-mean-squares estimation, and (4) a single element set comprised of a 2 order bivariate function.

Here we want to present our subfamily (SON - Self Organizing Networks) of the SOS algorithm family, characterized differently by: (1) global optimization search, (2) arbitrary connectivity based on an arbitrary number of neuron inputs, (3) Structure Estimation Criteria (SEC) (a variation of Rissanen's [1983]. Minimum Description Length Criteria, extended to the hierarchical case), and, (4) for training speed, activation functions are restricted to be linear on the parameters and the output functions need to be invertible, no other restriction is imposed in kind or number. The particular algorithm presented here is called the Self Organizing

Neural Network (SONN) [Tenorio&Lee, 1988,1989; Tenorio 1990 a,b]. It was composed of: (1) a graph synthesis procedure based on Simulated Annealing [Kirkpatrick et al., 1983]; (2) two input neurons that are arbitrarily connected; (3) the Structure Estimation Criteria; and, (4) a set of all polynomials that are special cases of 2nd order bivariates and inclusive, followed or not by sigmoid functions. The SONN algorithm performs a search in the model space by the construction of hypersurfaces. A network of nodes, each node representing a hypersurface, is organized to be an approximate model of the real system. Below, the components of SONN are discussed.

## 2.2 THE ALGORITHM STRUCTURE

The mechanisms behind the algorithm works as follows. First, create a set of terminals which are the output of the nodes available for connection to other nodes. This set is initialized with the output of the input nodes; in other words, the input variables themselves. From this set, with uniform probability, select a subset (2 in our case) of terminals, and used them as inputs to the new node. To construct the new node, select all the function of the set of prototype functions (activation followed by output function), and evaluate the SEC using the terminals as inputs. Selecting the best function, test for the acceptance of that node according to the Simulated Annealing move acceptance criterion. If the new node is accepted, place its output in the set of terminals and iterate until the optimum model is found. The details of the algorithm can be found in [Tenorio&Lee, 1989].

### 2.2.1 The Prototype Functions

Consider the Mahalanobis distance:

$$y_j = \text{sig}\{(\mathbf{x} - \mu)\mathcal{X}^{-1}(\mathbf{x} - \mu)^t\} \qquad (1)$$

This distance can be rewritten as a second order function, whose parameters are the indirect representation of the covariance matrix $\mathcal{X}$ and the mean vector $\mu$. This function is linear in the parameters, which makes it easy to perform training, and it is the function with the smallest degree of non linearity; only simpler is the linear case. Interestingly enough, this is the same prototype function used in the GMDH algorithm to form the Ivahnenko polynomial for apparently completely different reasons. In the SONN, this function is taken to be 2-input and all its possible variations (32) by setting parameters to zero are included in the set of activation functions. This set combined with the output function (the identify or sigmoid), for the set of prototype functions, used by the algorithm in the node construction.

### 2.2.2 Evaluation of the Model Based on the MDL Criterion

The selection rule of the neuron transfer function was based on a modification of the Minimal Description Length (MDL) information criterion. In [Rissanen, 1978], the principle of minimal description for statistical estimation was developed. The reason for the choice of such a criterion is that, in general the accuracy of the model can increase at the expense of simplicity in the number of parameters. The

increase of complexity might also be accompanied by the overfitting of the model. To overcome this problem, the MDL provides a trade-off between the accuracy and the complexity of the model by including the structure estimation term of the final model. The final model (with the minimal MDL) is optimum in the sense of being a consistent estimate of the number of parameters while achieving the minimum error [Rissanen, 1980]. Given a sequence of observations $X_1, X_2, ..., X_N$ from the random variable X, the dominant term of the MDL in [Rissanen, 1978] is:

$$MDL = -\log f(x \mid \theta) + 0.5 \, k \log N \qquad (2)$$

where $f(x \mid \theta)$ is the estimated probability density function of the model, k is the number of parameters, and N is the number of observations. The first term is actually the negative of the maximum likelihood (ML) with respect to the estimated parameter. The second term describes the structure of the models and it is used as a penalty for the complexity of the model.

## 3. EXAMPLE - THE COCKTAIL PARTY PROBLEM

The Cocktail Party Problem is the name given to the phenomenon that people can understand and track speech in a noisy environment, even when the noise is being made by other speakers. A simpler version of this problem is presented here: a 4 khz channel is excited with male speech and modem data additively at the same time. The task presented to the network is to separate both signals.

To compare the accuracy of the signal separation between the SONN and the Back Propagation algorithms a normalized RMSE is used as a performance index:

$$\text{normalized RMSE} = \frac{\text{RMSE}}{\text{StandardDevision}} \qquad (3)$$

### 3.1. EXPERIMENTS WITH BACK PROPAGATION

In order to design a filter using Back Propagation for this task, several architectures were considered. Since the input and output to the problem are time series, and such architectures are static, modifications to the original paradigm is required to deal with the time dimension. Several proposals have been made in this respect: tapped delay filters, recurrent architectures, low pass filter transfer functions, modified discriminant functions, and self excitatory connections (see [Wah, Tenorio, Merha, and Fortes, 90] ). The best result for this task was achieved by two tapped delay lines in the input layer, one for the input signal, the other for the output signal. The network was trained to recognize the speech signal from the mixed signal. The mixed signal had a speech to modem data energy ratio of 4:1, or 2.5 dB.

The network was designed to be a feedforward with 42 inputs (21 delayed versions of the input signal, and similarly for the output signal), 15 hidden units, and a single output unit. The network was trained with a single phoneme, taking about

10 cpu-hours on a Sequent machine. The network when presented with the trained phoneme added to the modem data, produced a speech reconstructability error equal to a nRMSE of 0.910. Previously several different configurations of the network were tried as well as different network parameters, and signal ratios of 1:1; all with poor results. Few networks actually converged to a final solution. A major problem with the BP architecture is that it can perfectly filter the signal in the first few samples, just to later demonstrate increasing amounts of cumulative errors; this instability may be fruit of the recurring nature of the architecture, and suboptimal weight training (Figure 2). The difficulty in finding and fine tuning the architecture, the training convergence, and time requirements led us to later stop pursuing the design of these filters with Back Propagation strategies.

### 3.2. EXPERIMENTS WITH SONN

At that time, the SONN algorithm had been successfully used for identification and prediction tasks [Tenorio&Lee; 88,89,90]. To make the task more realistic with possible practical utilization of this filter (Data-Over-Voice Circuits), the energy ratio between the voice and the modem data was reduced to 1:1, or 0 dB. A tapped delay line containing 21 delayed versions of the mixed signal was presented to the algorithm. Two sets of prototype functions were used, and both contained the full set of 32 variations of 2nd order bivariates. The first set had the identity (SONN-I experiments) and the second had a sigmoid (SONN-SIG experiments) as the output function for each node.

SONN-I created 370 nodes, designing a final model with 5 nodes. The final symbolic transfer function which represents the closed form function of the network was extracted. Using a Gould Powernode 9080, this search took 98.6 sec, with an average of 3.75 nodes/sec. The final model had an nRMSE of 0.762 (Figure 3) for reconstructed speech with the same BP data; with 19 weights. Training with the modem signal led to nRMSE of 0.762 (Figure 4) for the BP data. A search using the SONN-SIG model was allowed to generate 1000 nodes, designing a final model with 5 nodes. With the same computer, the second search took 283.42 sec, with an average 3.5 nodes/sec. The final model had an nRMSE comparable to the SONN-I (better by 5-10%); with 20 weights. The main characteristics of both signals were captured, specially if one looks at the plots and notices the same order of non-linearity between the real and estimated signals (no over or under estimation). Because of the forgiving nature of the human speech perception, the voice after reconstruction, although sightly muffled, remains of good quality; and the reconstructed modem signal can be used to reconstruct the original digital message, without much further post processing. The SONN does not present cumulative errors during the reconstruction, and when test with different (unseen, from the same speaker) speech data, performed as well as with the test data. We have yet to fully explore the implication of that to different speakers and with speaker of different gender or language. These results will be reported elsewhere.

## 4. COMPARISON BETWEEN THE TWO ALGORITHMS

Below we outline the comparison between the two algorithms drawn from our experience with this signal separation problem.

## 4.1. ADVANTAGES

The following were advantages of the SONN approach over the BP paradigm. The most striking difference was found in the training times, and in the amount of data required for training. The BP required 42 inputs (memories), where as the SONN functioned with 21 inputs, actually using as few as 4 in the final model (input variable selection). The SONN removed the problem of model estimation and architecture design. The number of connections with the SONN models is as low as 8 for 20 weights (relevant connections), as compared with 645 connections and weights for the BP model. The accuracy and complexity of the model can be trade for learning time as in BP, but the models that were more accurate also required less parameters than BP. The networks are not required to be homogeneous, thus contributing to smaller models as well. Above all, the SONN can produce both the C code for the network as well as the sequence of individual node symbolic functions; the SONN-I can also produce the symbolic representation of the closed form function of he entire network.

## 4.2. DISADVANTAGES

Certain disadvantages of using self-organizing topology networks with stochastic optimization algorithms were also apparent. The learning time of the SONN is non deterministic, and depends on the model complexity and starting point. Those are characteristic of the Simulated Annealing (SA) algorithm. These disadvantages are also present in the BP approach for different reasons. The connectivity of the model is not known a priori, which does not permit hardware implementation algorithms with direct connectivity emulation. Because the SONN selects nodes from a growing set with uniform probability, the probability of choosing a pair of nodes decreases with the inverse of the square of the number of nodes. Thus algorithm effectiveness decreases with processing time. Careful plotting of the SEC, nRMSE, and complexity trajectories during training reveal that the first 10% of the processing time achieves 90% of the final steady state values. Biasing the node selection procedure might be an alternative to modify this behavior. Simulated Annealing also required parametric tuning of the algorithm by setting" the initial and final temperature, the duration of the search at each temperature and the temperature decay. Alternative algorithms such as A* might produce a better alternative to stochastic search algorithms.

## 5. CONCLUSION AND FUTURE WORK

In this study, we proposed a new approach for the signal separation filter design based on a flexible, self-organizi neural network (SONN) algorithm. The variable structure provides the oppo nity to search and construct the optimal model based on input-output observations. The hierarchical v ion of the MDL, lled the Structure Estimation Criteria, was used to guide trade-off betwe the model complexity and the accuracy of the estimation. The SONN approach demonstrates potential usefulness as a tool for non linear signal processing function design.

We would like to explore the use of high level knowledge for function selection

and connectivity. Also, the issues involving estimator and deterministic searches are still open. Currently we are exploring the use of SONN for digital circuit synthesis, and studying how close the architecture generated here can approach the design of natural structures when performing similar functions. More classification problems, and problems involving dynamical systems (adaptive control and signal processing) need to be explored to give us the experience needed to tackle the problems for which it was designed.

## 6.  NOTE

The results reported here were originally intended for two papers accepted for presentation at the NIPS'89. The organizing committee asked us to fuse the into a single presentation for organizational purposes. In the limited time and the small space allocated for the presentation of these results, we sought a compromise between the reporting of the results and the description and comments on our experience with the algorithm. The interested reader should look at the other references about the SONN listed here and forthcoming papers.

## REFERENCE

A. G. Ivakhnenko, (1971) "Polynomial Theory of Complex Systems," IEEE Trans. S.M.C, Vol. SMC-1, no.4, pp. 364-378, Oct.

J. J. Duffy and M. A. Franklin, (1975) "A Learning Identification Algorithm and its Application to an Environmental System," IEEE Trans. S. M. C., Vol. SMC-5, no. 2, pp. 226-240.

S. Ikeda, M. Ochiai and Y. Sawarogi, (1976) "Sequential GMDH Algorithm and its Application to River Flow Prediction," IEEE Trans S.M.C., Vol. SMC-6, no.7, pp. 473-479, July.

H. Tamura, T. Kondo, (1980) "Heuristics Free Group Method of Data Handling Algorithm of Generating Optimal Partial Polynomials with Application to Air Pollution Predication," Int. J. Systems Sci., 11,no.9, pp. 1095-1111.

J. Rissanen (1978) "Modeling by Shortest Data Description," Automatica, Vol.14, pp. 465-471.

J. Rissanen, (1980) "Consistent Order Estimation of Autoregression Processes by Shortest Description of Data," Analysis and Optimization of Stochastic System, Jacobs et al eds. NY Academic.

J. Rissanen, (1983) "A Universal Prior for Integers and Estimation by Minimum Description Length," Annuals of Statistics, Vol.11, no. 2, pp. 416-431.

S.Kirkpatrick, C.D. Gelatt, M.P. Vecchi, (1983) "Optimization by Simulated Annealing," Science, vol.220, pp. 671-680, May.

M. F. M. Tenorio and W.-T. Lee, (1988) "Self-Organizing Neural Network for the Identification Problem," *Advances in Neural Information Processing Systems I*, David S. Touretzky ed., pp. 57-64.

M. F. M. Tenorio and W.-T. Lee, (1989) "Self-Organizing Neural Network for the

Identification Problem," School of Electrical Engineering, Purdue University, Tech Report TR-EE 89-20, June.

M. F. M. Tenorio and W. -T. Lee, (1990) "Self-Organizing Network for the Identification Problem," (expanded) IEEE Trans. on the Neural Networks, to appear.

M. F. M. Tenorio, (1990) "The Self-Organizing Neural Network Algorithm: Adapting Topology for Optimum Supervised Learning," IEEE Hawaii Conference in Systems Science, 22, January.

M. F. Tenorio, (1990) "Self-Organizing Neural Network for the Signal Separation Problem," to be submitted.

B. Wah, M. Tenorio, P. Mehra, J. Fortes, (1990) *Artificial Neural Networks: Theory, Algorithms, Application and Implementations,*" IEEE press.

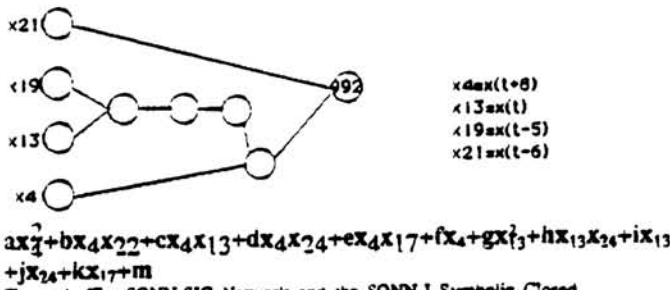

$$ax_4^2+bx_4x_{22}+cx_4x_{13}+dx_4x_{24}+ex_4x_{17}+fx_4+gx_{13}^2+hx_{13}x_{24}+ix_{13}$$
$$+jx_{24}+kx_{17}+m$$

Figure 1: The SONN-SIG Network and the SONN-I Symbolic Closed Form

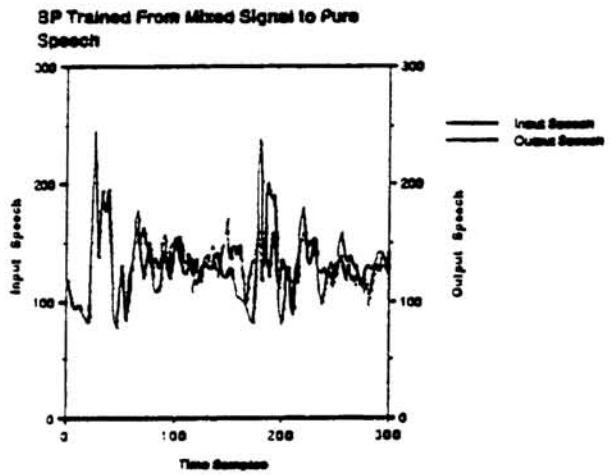

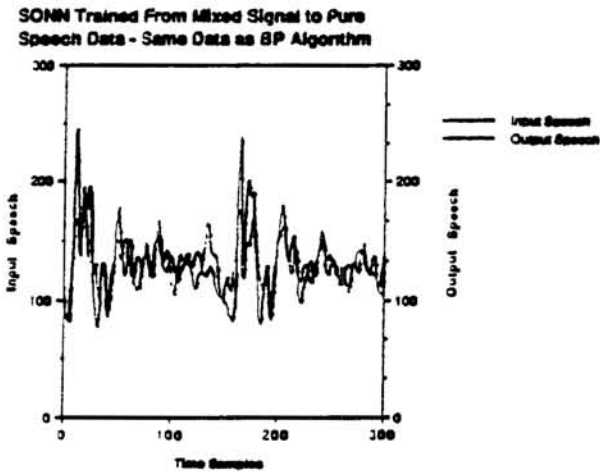

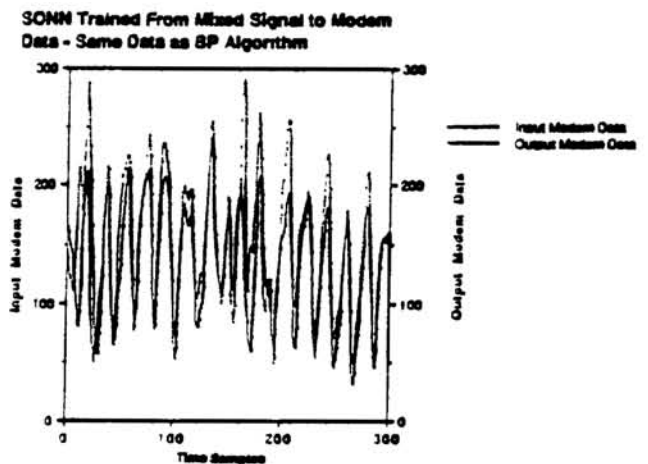